# Development and Spatial Structure of Cortical Feature Maps: A Model Study

**K. Obermayer**
Beckman-Institute
University of Illinois
Urbana, IL 61801

**H. Ritter**
Technische Fakultät
Universität Bielefeld
D-4800 Bielefeld

**K. Schulten**
Beckman-Institute
University of Illinois
Urbana, IL 61801

## Abstract

Feature selective cells in the primary visual cortex of several species are organized in hierarchical topographic maps of stimulus features like "position in visual space", "orientation" and "ocular dominance". In order to understand and describe their spatial structure and their development, we investigate a self-organizing neural network model based on the feature map algorithm. The model explains map formation as a dimension-reducing mapping from a high-dimensional feature space onto a two-dimensional lattice, such that "similarity" between features (or feature combinations) is translated into "spatial proximity" between the corresponding feature selective cells. The model is able to reproduce several aspects of the spatial structure of cortical maps in the visual cortex.

## 1 Introduction

Cortical maps are functionally defined structures of the cortex, which are characterized by an ordered spatial distribution of functionally specialized cells along the cortical surface. In the primary visual area(s) the response properties of these cells must be described by several independent features, and there is a strong tendency to map combinations of these features onto the cortical surface in a way that translates "similarity" into "spatial proximity" of the corresponding feature selective cells (see e.g. [1-6]). A neighborhood preserving mapping between a high-dimensional feature space and the two dimensional cortical surface, however, cannot be achieved, so the spatial structure of these maps is a compromise, preserving some neighborhood relations at the expense of others.

The compromise realized in the primary visual area(s) is a hierarchical representation of features. The variation of the *secondary* features "preferred orientation",

"orientation specifity" and "ocular dominance" is highly repetitive across the *primary* map of retinal location, giving rise to a large number of small maps, each containing a complete representation of the full range of the secondary features. If the neighborhood relations in feature space are to be preserved and maps must be continuous, the spatial distributions of the secondary features "orientation preference", "orientation specifity" and "ocular dominance" can no longer be independent. Interestingly, there is experimental evidence in the macaque that this is the case, namely, that regions with smooth change in one feature (e.g. "ocular dominance") correlate with regions of rapid change in another feature (e.g. "orientation") [7,8]. Preliminary results [9] indicate that these correlations may be a natural consequence of a dimension reducing mapping which preserves neighborhood relations.

In a previous study, we investigated a model for the joint formation of a retinotopic projection and an orientation column system [10], which is based on the self-organizing feature map algorithm [11,12]. This algorithm generates a representation of a given manifold in feature space on a neural network with prespecified topology (in our case a two-dimensional sheet), such that the mapping is continous, smooth and neighborhood relations are preserved to a large extent.[1] The model has the advantage that its rules can be derived from biologically plausible developmental principles [15,16]. Therefore, it can be interpreted not only as a pattern model, which generates a representation of feature combinations subject to a set of constraints, but also as a pattern formation model, which describes an input driven developmental process. In this contribution we will extend our previous work by the addition of another secondary feature, "ocular dominance" and we will concentrate on the hierarchical mapping of feature combinations as a function of the set of input patterns.

## 2    Description of the Model

In our model the cortical surface is divided into $N \times N$ small patches, *units $\vec{r}$*, which are arranged on a two-dimensional lattice (network layer) with periodic boundary conditions (to avoid edge effects). The functional properties of neurons located in each patch are characterized by a *feature vector $\vec{w}_{\vec{r}}$*, which is associated with each unit $\vec{r}$ and whose components $(\vec{w}_{\vec{r}})_k$ are interpreted as receptive field properties of these neurons. The feature vectors, $\vec{w}_{\vec{r}}$, as a function of unit locations $\vec{r}$, describe the spatial distribution of feature selective cells over the cortical layer, i.e. the cortical map.

To generate a representation of features along the network layer, we use the self-organizing feature map algorithm [1,2]. This algorithm follows an iterative procedure. At each step an *input vector $\vec{v}$*, which is of the same dimensionality as $\vec{w}_{\vec{r}}$, is chosen at random according to a probability distribution $P(\vec{v})$. Then the unit $\vec{s}$, whose feature vector $\vec{w}_{\vec{s}}$ is closest to the input pattern $\vec{v}$, is selected and the components $(\vec{w}_{\vec{r}})_k$ of it's feature vector are changed according to the feature map learning rule:

$$\vec{w}_{\vec{r}}(t+1) \; = \; \vec{w}_{\vec{r}}(t) \; + \; \varepsilon(t)h(\vec{r},\vec{s},t)(\vec{v}-\vec{w}_{\vec{r}}(t)) \tag{1}$$

where $h(\vec{r}, \vec{s}, t)$, the *neighborhood function*, is given by:

$$h(\vec{r}, \vec{s}, t) = \exp\left(-(r_1 - s_1)^2/\sigma_{h1}^2(t) - (r_2 - s_2)^2/\sigma_{h2}^2(t)\right). \tag{2}$$

## 3   Coding of Receptive Field Properties

In the following we describe the receptive field properties by the feature vector $\vec{w}_{\vec{r}}$ given by $\vec{w}_{\vec{r}} = (x_{\vec{r}}, \; y_{\vec{r}}, \; q_{\vec{r}}\cos(2\phi_{\vec{r}}), \; q_{\vec{r}}\sin(2\phi_{\vec{r}}), \; z_{\vec{r}})$ where $(x_{\vec{r}}, \; y_{\vec{r}})$ denotes the position of the receptive field centers in visual space, $(\phi_{\vec{r}})$ the preferred orientation, and $(q_{\vec{r}})$, $(z_{\vec{r}})$ two quantities, which qualitatively can be interpreted as orientation specificity (see e.g. [17]) and ocular dominance (see e.g. [18]). If $q_{\vec{r}}$ is zero, then the units are unspecific for orientation; the larger $q_{\vec{r}}$ becomes, the sharper the units are tuned. "Binocular" units are characterized by $z_{\vec{r}} = 0$, "monocular" units by a large positive or negative value of $z_{\vec{r}}$. "Similarity" between receptive field properties is then given by the euclidean distance between the corresponding feature vectors.

The components $\vec{w}_{\vec{r}}$ of the input vector $\vec{v} = (x, \; y, \; q\cos(2\phi), \; q\sin(2\phi), \; z)$ describe stimulus features which should be represented by the cells in the cortical map. They denote position in the visual field $(x, \; y)$, orientation $\phi$, and two quantities $q$ and $z$ qualitatively describing pattern eccentricity and the distribution of activity between both eyes, respectively. Round stimuli are characterized by $q = 0$ and the more eliptic a pattern is the larger is the value of $q$. A "binocular" stimulus is characterized by $z = 0$, while a "monocular" stimulus is characterized by a large positive or negative value of $z$ for "right eye" or "left eye" preferred, respectively.

Input vectors were chosen with equal probability from the manifold

$$V = \{\vec{v} \mid x, y \; \epsilon \; [0, d]; \; \phi \; \epsilon [0, \pi]; \; q = q_{pat}; \; |z| = z_{pat}\}, \tag{3}$$

i.e. all feature combinations characterized by a fixed value of $q$ and $|z|$ were selected equally often. If the model is interpreted from a developmental point of view, the manifold $V$ describes properties of (subcortical) activity patterns, which drive map formation. The quantities $d$, $q_{pat}$ and $z_{pat}$ determine the feature combinations to be represented by the map. As we will see below, their values crucially influence the spatial structure of the feature map.

## 4   Hierarchical Maps

If $q_{pat}$ and $z_{pat}$ are smaller than a certain threshold then "orientation preference", "orientation selectivity" and "ocular dominance" are *not* represented in the map (i.e. $q_{\vec{r}} = z_{\vec{r}} = 0$) but fluctuate around a stationary state of eq. (1), which corresponds to a perfect topographic representation of visual space. In this parameter regime, the requirement of a continous dimension-reducing map leads to the suppression of the additional features "orientation" and "ocular dominance".

Let us consider an ensemble of networks, each characterized by a set $\{\vec{w}_{\vec{r}}\}$ of feature vectors, and denote the time-dependent distribution function of this ensemble by

$S(\vec{w},t)$. Following a method derived in [19], we can describe the time-development of $S(\vec{w},t)$ near the stationary state by the Fokker-Planck equation

$$\frac{1}{\epsilon}\partial_t S(\{\vec{u}_{\vec{r}}\},t) = \sum_{\vec{p}m\vec{q}n} \frac{\partial}{\partial \vec{u}_{\vec{p}m}} B_{\vec{p}m\vec{q}n} \vec{u}_{\vec{q}n} S(\{\vec{u}_{\vec{r}}\},t) + \frac{\epsilon}{2} \sum_{\vec{p}m\vec{q}n} D_{\vec{p}m\vec{q}n} \frac{\partial^2 S(\{\vec{u}_{\vec{r}}\},t)}{\partial \vec{u}_{\vec{p}m} \partial \vec{u}_{\vec{q}n}} \quad (4)$$

where the origin of $S(.,t)$ was shifted to the stationary state $\{\vec{w}_{\vec{r}}\}$, using now the new argument variable $\vec{u}_{\vec{r}} = \vec{w}_{\vec{r}} - \vec{\bar{w}}_{\vec{r}}$. The eigenvalues of $\underline{B}$ determine the stability of the stationary state, the topographic representation, while $\underline{B}$ and $\underline{D}$ together govern size and time development of fluctuations $< u_{\vec{p}i} u_{\vec{q}j} >$.

Let us define the Fourier modes $\vec{u}_{\vec{k}}$ of the equilibrium deviations $\vec{u}_{\vec{r}}$ by $\hat{\vec{u}}_{\vec{k}} = 1/N \sum_{\vec{r}} e^{i\vec{k}\vec{r}} \vec{u}_{\vec{r}}$. For small values of $q_{pat}$ and $z_{pat}$ the eigenvalues of $\underline{B}$ are all negative, hence the topographic stationary state is stable. If $q_{pat}$ and $z_{pat}$ are larger than[2]

$$q_{thres} = \sqrt{\frac{e}{2}} \frac{d}{N} \min(\sigma_{h1}, \sigma_{h2}), \quad z_{thres} = \frac{1}{2}\sqrt{e} \frac{d}{N} \min(\sigma_{h1}, \sigma_{h2}), \quad (5)$$

however, the eigenvalues corresponding to the set of modes $\hat{\vec{u}}_{\vec{k}}$ which are perpendicular to the $(x,y)$-plane and whose wave-vectors $\vec{k}$ are given by

$$|\vec{k}| = 2/\sigma_h \; if \; \sigma_{h1} = \sigma_{h2}, \quad \left. \begin{array}{ccc} k_x & = & \pm 2/\sigma_{h1} \\ k_y & = & 0 \end{array} \right\} \; if \; \sigma_{h1} < \sigma_{h2}, \quad (6)$$

become positive. For larger values of $q_{pat}$ and $z_{pat}$ then, the topographic state becomes unstable and a "column system" forms.

For an isotropic neighborhood function ($\sigma_{h1} = \sigma_{h2} = \sigma_h$), the matrices $\hat{B}(\vec{k})$ and $\hat{D}(\vec{k})$ can be diagonalized simultaneously and the mean square amplitude of the fluctuations around the stationary state can be given in explicit form:

$$< u_{\|}^2(\vec{k}) > = \pi\frac{\epsilon}{2}\sigma_h^2 \frac{d^2}{N^2} \frac{(\sigma_h^4 k^2/4 + 1/12)\exp(-\sigma_h^2 k^2/4)}{\exp(\sigma_h^2 k^2/4) - 1 + \sigma_h^2 k^2/2} \quad (7)$$

$$< u_{\perp}^2(\vec{k}) > = \pi\frac{\epsilon}{24}\sigma_h^2 \frac{d^2}{N^2} \frac{\exp(-\sigma_h^2 k^2/4)}{\exp(\sigma_h^2 k^2/4) - 1} \quad (8)$$

$$< u_{y1}^2(\vec{k}) > = < u_{y2}^2(\vec{k}) > = \pi\frac{\epsilon}{4}\sigma_h^2 q_{pat}^2 \frac{\exp(-\sigma_h^2 k^2/4)}{\exp(\sigma_h^2 k^2/4) - (N^2 q_{pat}^2 k^2)/(2d^2)} \quad (9)$$

$$< u_z^2(\vec{k}) > = \pi\frac{\epsilon}{2}\sigma_h^2 z_{pat}^2 \frac{\exp(-\sigma_h^2 k^2/4)}{\exp(\sigma_h^2 k^2/4) - (N^2 q_{pat}^2 k^2)/d^2} \quad (10)$$

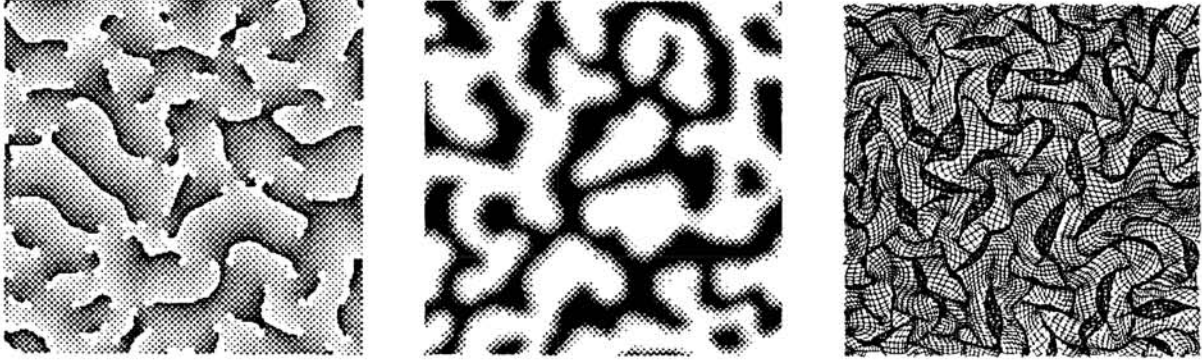

Figure 1: "Orientation preference" (a, left), "ocular dominance" (b, center) and locations of receptive field centers (c, right) as a function of unit loaction. Figure 1a displays an enlarged section of the "orientation map" only. Parameters of the simulation were: $N = 256$, $d = 256$, $q_{pat} = 12$, $z_{pat} = 12$, $\sigma_h = 5$, $\varepsilon = 0.02$

where $u_{\parallel}$, $u_{\perp}$ denote the amplitude of fluctuations parallel and orthogonal to $\vec{k}$ in the $(x, y)$-plane, $u_{y1}$, $u_{y2}$ parallel to the orientation feature dimension and $u_z$ parallel to the ocular dominance feature dimension, respectively.

Thus, for $q_{pat} \to q_{thres}$ or $z_{pat} \to z_{thres}$ the mean square amplitudes of fluctuations diverge for the modes which become unstable at the threshold (the denominator of eqs. (9,10) approaches zero) and the relaxation time of these fluctuations goes to infinity (not shown). The fact that either a ring or two groups of modes become unstable is reflected in the spatial structure of the maps above threshold.

For larger values of $q_{pat}$ and $z_{pat}$ orientation and ocular dominance are represented by the network layer, i.e. feature values fluctuate around a stationary state which is characterized by a certain distribution of feature-selective cells. Figure 1 displays orientation preference $\phi_{\vec{r}}$ (Fig. 1a), ocular dominance $z_{\vec{r}}$ (Fig. 1b) and the locations $(x_{\vec{r}}, y_{\vec{r}})$ of receptive field centers in visual space (Fig. 1c) as a function of unit location $\vec{r}$. Each pixel of the images in Figs. 1a,b corresponds to a network unit $\vec{r}$. Feature values are indicated by gray values: black $\to$ white corresponds to an angle of $0^\circ \to 180^\circ$ (Fig. 1a) and to an ocular dominance value of $0 \to$ max (Fig. 1b). White dots in Fig. 1a mark regions where units still completely unspecific for orientation are located ("foci"). In Fig. 1c the receptive field center of every unit is marked by a dot. The centers of units which are neighbors in the network layer were connected by lines, which gives rise to the net-like structure.

The overall preservation of the lattice topology, and the absence of any larger discontinuities in Fig. 1c, demonstrate that "position" plays the role of the primary stimulus variable and varies in a topographic fashion across the network layer. On a smaller length scale, however, numerous distortions are visible which are caused by the representation of the other features, "orientation" and "ocular dominance". The variation of these secondary features is highly repetitive and patterns strongly resembling orientation columns (Fig. 1b) and ocular dominance stripes (Fig. 1c) have formed. Note that regions unspecific for orientation as well as "binocular" regions exist in the final map, although these feature combinations were not present in the set of input patterns (3). They are correlated with regions of high magnitude of the "orientation" and "ocular dominance"-gradients, respectively (not shown). These structures are a consequence of the neighborhood preserving and dimension

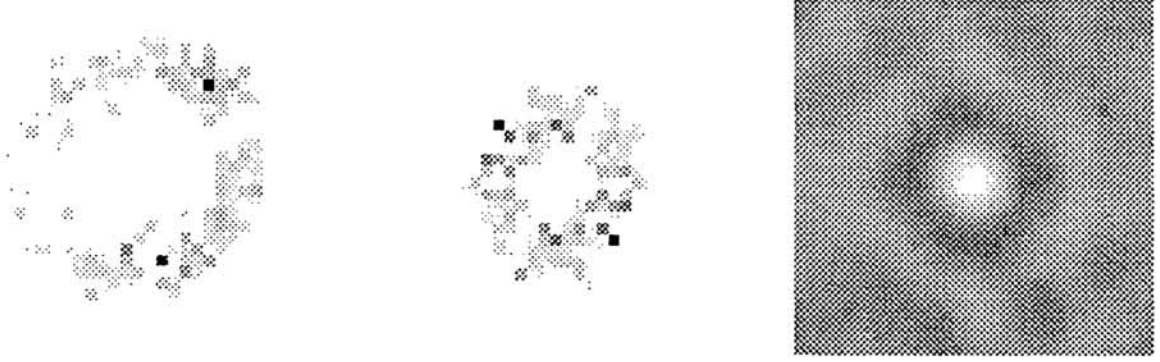

Figure 2: Two-dimensional Fourier spectra of the "orientation" (a, left) and "ocular dominance" (b, center) coordinates for the map shown in Fig. 1. c, right: Autocorrelation function of the feature coordinate $w_{\vec{r}3}$ for the map shown in Fig. 1.

reducing mapping; they do not result from the requirement of representing this particular set of feature combinations.[3]

Figure 2a,b shows the two-dimensional Fourier spectra $\hat{w}_{\vec{k},occ} = \sum_{\vec{r}} e^{i\vec{k}\vec{r}} z_{\vec{r}}$ and $\hat{w}_{\vec{k},ori} = \sum_{\vec{r}} e^{i\vec{k}\vec{r}} q_{\vec{r}}(\cos(2\phi_{\vec{r}}) + i\sin(2\phi_{\vec{r}}))$ for the "ocular dominance" (Fig. 2b) and "orientation" (Fig. 2a) coordinates, respectively. Each pixel corresponds to a single mode $\vec{k}$ and its brightness indicates the mean square amplitude $|\hat{w}_{\vec{k}}|^2$ of the mode $\vec{k}$. For an isotropic neighborhood function the orientation map is characterized by wave vectors from a ring shaped region in the Fourier domain (Fig. 2a), which becomes eccentric with increasing $\sigma_{h1}/\sigma_{h2}$ (not shown) until the ring dissolves into two separate groups of modes. The phases (not shown) seem to be random, but we cannot exclude correlations completely. Figure 2c shows the autocorrelation function $S_{33}(\vec{s}) = < w_{(\vec{r}-\vec{s})3}\, w_{(\vec{s})3} >$ as a function of the distance $\vec{s}$ between cells in the network layer. The origin of the $\vec{s}$-plane is located in the center of the image and the brightness indicates a positive (white), zero (medium gray) or negative (black) value of $S_{33}$. The autocorrelation functions have a Mexican-hat form. The (negative) minimum is located at half the wavelength $\lambda$ associated with the the wave number $|\vec{k}|$ of the modes with high amplitude in Fig. 2a. At this distance the response properties of the units are anticorrelated to some extent. If cells are separated by a distance larger than $\lambda$, the response properties are uncorrelated.

If $q_{pat}$ and $z_{pat}$ are large enough, the feature hierarchy observed in Figs. 1,2 breaks down and "preferred orientation" or "ocular dominance" plays the role of the primary stimulus variable. Figure 3 displays orientation preference $\phi_{\vec{r}}$ (Fig. 3a) and ocular dominance $z_{\vec{r}}$ (Fig. 3b) as a function of unit location $\vec{r}$. There is only one continous region for each interval of "preferred orientation" and for each eye, but each of these regions now contains a representation of a large part of visual space. Consequently the position map shows multiple representations of visual space.

Hierarchical maps are generated by the feature map algorithm whenever there is a hierarchy in the variances of the set of patterns along the various feature dimensions

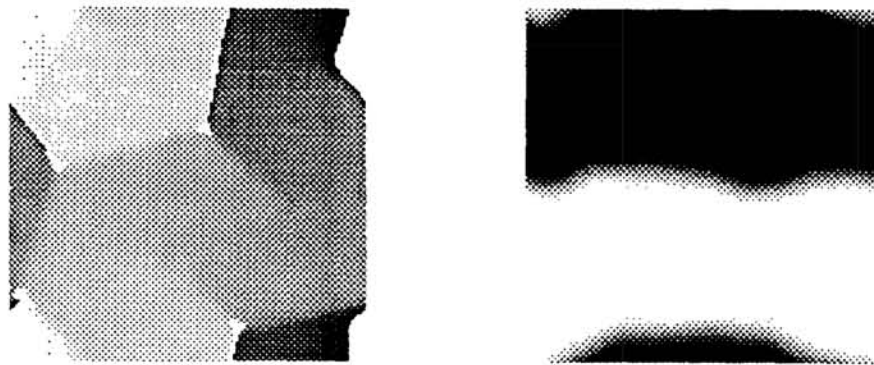

Figure 3: "Orientation preference" (a, left) and "ocular dominance" (b, center) as a function of unit loaction for a map generated using a large value of $q_{pat}$ and $z_{pat}$. Parameters were: $N = 128$, $d = 128$, $q_{pat} = 2500$, $z_{pat} = 2500$, $\sigma_h = 5$, $\varepsilon = 0.1$

(In our example a hierarchy in the magnitudes of $d$, $q_{pat}$ and $z_{pat}$). The features with the largest variance become the primary feature; the other features become secondary features, which are represented multiple times on the network layer.

## Acknowledgements

The authors would like to thank the Boehringer-Ingelheim Fonds for financial support by a scholarship to K. O. This research has been supported by the National Science Foundation (grant number 9017051). Computer time on the Connection Machine CM-2 has been made available by the National Center for Supercomputer Applications at Urbana-Champaign and the Pittsburgh Supercomputing Center both supported by the National Science Foundation.

## Footnotes

[1] For other modelling approaches along these lines see [13,14].

[2] In the derivation of the following formulas several approximations have to be made. A comparison with numerical simulations, however, demonstrate that these approximations are valid except if the value $q_{pat}$ or $z_{pat}$ is within a few percent of $q_{thres}$ or $z_{thres}$, respectively. Details of these calculations will be published elsewhere

[3]In the cortex, however, cells unspecific for orientation seem to be important for visual processing. To improve the description of the spatial structure of cortical maps, it is necessary to include these feature combinations into the set $V$ of input patterns (see [9]).

## References

[1] Hubel D.H. and Wiesel T.N. (1974), J. Comp. Neurol. **158**, 267-294
[2] Blasdel G.G. and Salama G. (1986), Nature **321**, 579-585
[3] Grinvald A. et al. (1986), Nature **324**, 361-364
[4] Swindale N.V. et al. (1987), J. Neurosci. **7**, 1414-1427
[5] Löwel S. et al. (1987), **255**, 401-415
[6] Ts'o D.Y. et al., Science **249**, 417-420
[7] Livingstone M.S. and Hubel D.H. (1984), J. Neurosci. **4**, 309-356
[8] Blasdel G.G. (1991), in preparation
[9] Obermayer K. et al. (1991), Proc. of the ICANN-91, Helsinki, submitted
[10] Obermayer K. et al. (1990), Proc. Natl. Acad. Sci. USA **87**, 8345-8349
[11] Kohonen T. (1982a), Biol. Cybern. **43**, 59-69
[12] Kohonen T. (1982b), Biol. Cybern.**44**, 135-140
[13] Nelson M.E. and Bower J.M. (1990), TINS **13**, 401-406
[14] Durbin R. and Mitchison M. (1990), Nature **343**, 644-647
[15] von der Malsburg C. (1973), Kybernetik **14**, 85-100
[16] Kohonen T. (1983), Self-Organization and Associative Memory, Springer-Verlag, New York
[17] Swindale N.V. (1982), Proc. R. Soc. Lond., **B215**, 211-230
[18] Goodhill G.J. and Willshaw D.J. (1990), Network **1**, 41-59
[19] Ritter H. and Schulten K. (1989), Biol. Cybern. **60**, 59-71
